# Capacity and Information Efficiency of a Brain-like Associative Net

**Bruce Graham and David Willshaw**
Centre for Cognitive Science, University of Edinburgh
2 Buccleuch Place, Edinburgh, EH8 9LW, UK
Email: bruce@cns.ed.ac.uk & david@cns.ed.ac.uk

## Abstract

We have determined the capacity and information efficiency of an associative net configured in a brain-like way with partial connectivity and noisy input cues. Recall theory was used to calculate the capacity when pattern recall is achieved using a *winners-take-all* strategy. Transforming the *dendritic sum* according to *input activity* and *unit usage* can greatly increase the capacity of the associative net under these conditions. For moderately sparse patterns, maximum information efficiency is achieved with very low connectivity levels ($\leq 10\%$). This corresponds to the level of connectivity commonly seen in the brain and invites speculation that the brain is connected in the most information efficient way.

## 1  INTRODUCTION

Standard network associative memories become more plausible as models of associative memory in the brain if they incorporate (1) partial connectivity, (2) sparse activity and (3) recall from noisy cues. In this paper we consider the capacity of a binary associative net (Willshaw, Buneman, & Longuet-Higgins, 1969; Willshaw, 1971; Buckingham, 1991) containing these features. While the associative net is a very simple model of associative memory, its behaviour as a storage device is not trivial and yet it is tractable to theoretical analysis. We are able to calculate

the capacity of the net in different configurations and with different pattern recall strategies. Here we consider the capacity as a function of connectivity level when *winners-take-all recall* is used.

The associative net is an heteroassociative memory in which pairs of binary patterns are stored by altering the connection weights between input and output units via a Hebbian learning rule. After pattern storage, an output pattern is recalled by presenting a previously stored input pattern on the input units. Which output units become active during recall is determined by applying a threshold of activation to measurements that each output unit makes of the input cue pattern. The most commonly used measurement is the weighted sum of the inputs, or *dendritic sum*. Amongst the simpler thresholding strategies is the *winners-take-all (WTA)* approach, which chooses the required number of output units with the highest dendritic sums to be active. This works well when the net is fully connected (each input unit is connected to every output unit), and input cues are noise-free. However, recall performance deteriorates rapidly if the net is partially connected (each input unit is connected to only some of the output units) and cues are noisy.

Marr (1971) recognised that when an associative net is only partially connected, another useful measurement for threshold setting is the total *input activity* (sum of the inputs, regardless of the connection weights). The ratio of the dendritic sum to the input activity can be a better discriminator of which output units should be active than the dendritic sum alone. Buckingham and Willshaw (1993) showed that differences in *unit usage* (the number of patterns in which an output unit is active during storage) causes variations in the dendritic sums that makes accurate recall difficult when the input cues are noisy. They incorporated both input activity and unit usage measurements into a recall strategy that minimised the number of errors in the output pattern by setting the activity threshold on a unit by unit basis. This is a rather more complex threshold setting strategy than a simple winners-take-all.

We have previously demonstrated via computer simulations (Graham & Willshaw, 1994) that the *WTA* threshold strategy can achieve the same recall performance as this minimisation approach if the dendritic sums are transformed by certain functions of the input activity and unit usage before a threshold is applied. Here we calculate the capacity of the associative net when *WTA* recall is used with three different functions of the dendritic sums: (1) pure dendritic sums, (2) modified by input activity and (3) modified by input activity and unit usage. The results show that up to four times the capacity can be obtained by transforming the dendritic sums by a function of both input activity and unit usage. This increase in capacity was obtained without a loss of information efficiency. For the moderately sparse patterns used, *WTA* recall is most information efficient at low levels of connectivity ($\leq 10\%$), as is the minimisation approach to threshold setting (Buckingham, 1991). This connectivity range is similar to that commonly seen in the brain.

## 2 NOTATION AND OPERATION

The associative net consists of $N_B$ binary output units each connected to a proportion $Z$ of the $N_A$ binary input units. Pairs of binary patterns are stored in the net. Input and output patterns contain $M_A$ and $M_B$ active units, respectively (activity level $\alpha = M/N \ll 1$). All connection weights start at zero. On presentation to the net of a pattern pair for storage, the connection weight between an active input unit and an active output unit is set to 1. During recall an input cue pattern is presented on the input units. The input cue is a noisy version of a previously stored input pattern in which a fraction, $s$, of the $M_A$ active units do not come from the stored pattern. A thresholding strategy is applied to the output units to determine which of them should be active. Those that should be active in response to the input cue will be called *high* units, and those that should be inactive will be called *low* units. We consider *winners-take-all (WTA)* thresholding strategies that choose to be active the $M_B$ output units with the highest values of three functions of the dendritic sum, $d$, the input activity, $a$, and the unit usage, $r$. These functions are listed in Table 1. The *normalised* strategy deals with partial connectivity. The *transformed* strategy reduces variations in the dendritic sums due to differences in unit usage. This function minimises the variance of the *low* unit dendritic sums with respect to the unit usage (Graham & Willshaw, 1994).

Table 1: WTA Strategies

| WTA Strategy | Function |
|---|---|
| Basic | $d$ |
| Normalised | $d' = d/a$ |
| Transformed | $d^* = 1 - (1 - d/a)^{1/r}$ |

## 3 RECALL THEORY

The capacity of the associative net is defined to be the number of pattern pairs that can be stored before there is one bit in error in a recalled output pattern. This cannot be calculated analytically for the net configuration under study. However, it can be determined numerically for the *WTA* recall strategy by calculating the recall response for different numbers of stored patterns, $R$, until the minimum value of $R$ is found for which a recall error occurs. The *WTA* recall response can be calculated theoretically using expressions for the distributions of the dendritic sums of *low* and *high* output units. The probability that the dendritic sum of a *low* or *high* output unit should have a particular value $x$ is, respectively (Buckingham & Willshaw, 1993; Buckingham, 1991)

$$P(d_l = x) = \sum_{r=1}^{R} \binom{R}{r} \alpha_B^r (1-\alpha_B)^{R-r} \binom{M_A}{x} (Z\rho[r])^x (1 - Z\rho[r])^{M_A - x} \quad (1)$$

$$P(d_h = x) = \sum_{r=0}^{R-1} \binom{R}{r} \alpha_B^r (1-\alpha_B)^{R-r} \binom{M_A}{x} (Z\mu[r+1])^x (1-Z\mu[r+1])^{M_A-x}$$

$$(2)$$

where $\rho[r]$ and $\mu[r]$ are the probabilities that an arbitrarily selected active input is on a connection with weight 1. For a *low* unit, $\rho[r] = 1 - (1-\alpha_A)^r$. For a *high* unit a good approximation for $\mu$ is $\mu[r+1] \simeq g + s\rho[r] = 1 - s(1-\alpha_A)^r$ where $g$ and $s$ are the probabilities that a particular active input in the cue pattern is *genuine* (belongs to the stored pattern) or *spurious*, respectively ($g + s = 1$) (Buckingham & Willshaw, 1993). The *basic WTA* response is calculated using these distributions by finding the threshold, $T$, that gives

$$(N_B - M_B)P(d_l \geq T) + M_B P(d_h \geq T) = M_B \qquad (3)$$

The number of false positive and false negative errors of the response is given by

$$E = (N_B - M_B)P(d_l \geq T) + M_B(1 - P(d_h \geq T)) \qquad (4)$$

The actual distributions of the *normalised* dendritic sums are the distributions of $d/a$. For the purposes of calculating the *normalised WTA* response, it is possible to use the *basic* distributions for the situation where every unit has the mean input activity, $a_m = M_A Z$. In this case the *low* and *high* unit distributions are approximately

$$P(d_l' = x) = \sum_{r=1}^{R} \binom{R}{r} \alpha_B^r (1-\alpha_B)^{R-r} \binom{a_m}{x} (\rho[r])^x (1-\rho[r])^{a_m-x} \qquad (5)$$

$$P(d_h' = x) = \sum_{r=0}^{R-1} \binom{R}{r} \alpha_B^r (1-\alpha_B)^{R-r} \binom{a_m}{x} (\mu[r+1])^x (1-\mu[r+1])^{a_m-x} \qquad (6)$$

Due to the nonlinear transformation used, it is not possible to calculate the *transformed* distributions as simple sums of binomials, so the following approach is used to generate the *transformed WTA* response. For a given *transformed* threshold, $T^*$, and for each possible value of unit usage, $r$, an equivalent *normalised* threshold is calculated via

$$T'[r] = a_m(1 - (1-T^*)^r) \qquad (7)$$

The *transformed* cumulative probabilities can then be calculated from the *normalised* distributions:

$$P(d_l^* \geq T^*) = \sum_{r=1}^{R} \binom{R}{r} \alpha_B^r (1-\alpha_B)^{R-r} P(d_l' \geq T'[r]) \qquad (8)$$

$$P(d_h^* \geq T^*) = \sum_{r=0}^{R-1} \binom{R}{r} \alpha_B^r (1-\alpha_B)^{R-r} P(d_h' \geq T'[r+1]) \qquad (9)$$

The *normalised* and *transformed* WTA responses are calculated in the same manner as the *basic* response, using the appropriate probability distributions.

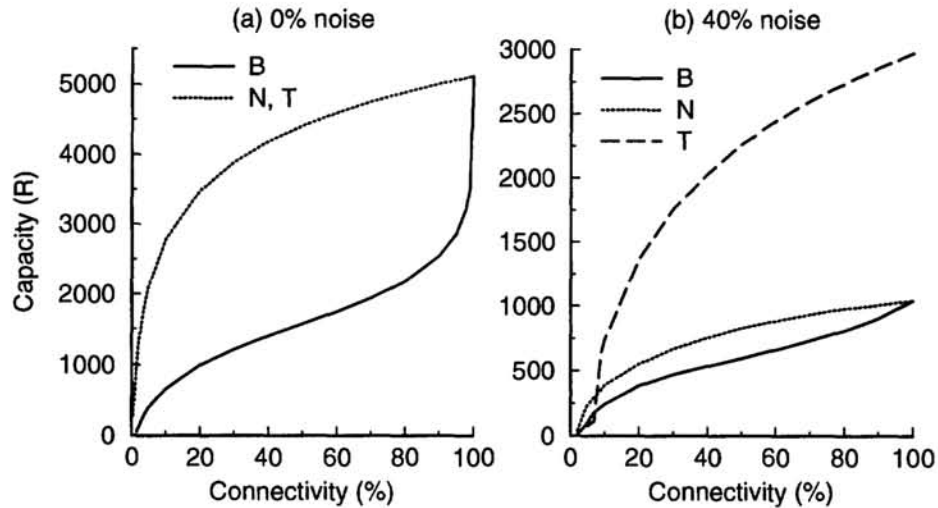

Figure 1: Capacity Versus Connectivity

## 4 RESULTS

Extensive simulations were previously carried out of *WTA* recall from a large associative net with the following specifications (Graham & Willshaw, 1994): $N_A = 48000$, $M_A = 1440$, $N_B = 6144$, $M_B = 180$. Agreement between the simulations and the theoretical recall described above is extremely good, indicating that the approximations used in the theory are valid. Here we use the theoretical recall to calculate capacity results for this large associative net that are not easily obtained via simulations. All the results shown have been generated using the theory described in the previous section.

Figure 1 shows the capacity as a function of connectivity for the different *WTA* strategies when there is no noise in the input cue, or 40% noise in the cue (legend: B = *basic WTA*, N = *normalised WTA*, T = *transformed WTA*; for clarity, individual data points are omitted). With no noise in the cue the *normalised* and *transformed* methods perform identically, so only the *normalised* results are shown. Figure 1(a) highlights the effectiveness of normalising the dendritic sums against input activity when the net is partially connected. Figure 1(b) shows the effect of noise on capacity. The capacity of each recall strategy at a given connectivity level is much reduced compared to the noise-free case. However, for connectivities greater than 10% the capacity of the *transformed WTA* is now much greater than either the *normalised* or *basic WTA*.

The relative capacities of the different strategies are shown in Figure 2 (legend: $N/B$ = ratio of *normalised* to *basic* capacity, $T/B$ = ratio of *transformed* to *basic*, $T/N$ = ratio of *transformed* to *normalised*). In the noise-free case (Figure 2(a)), at low levels of connectivity the relative capacity is distorted because the *basic* capacity

drops to near zero, so that even low *normalised* capacities are relatively very large. For most connectivity levels (10-90%) the *normalised WTA* provides 2-4 times the capacity of the *basic WTA*. In the noisy case (Figure 2(b)), the *normalised* capacity is only up to 1.5 times the *basic* capacity over this range of connectivities. The *transformed WTA*, however, provides 3 to nearly 4 times the *basic* capacity and 2.5 to nearly 3 times the *normalised* capacity for connectivities greater than 10%.

The capacities can be interpreted in information theoretic terms by considering the information efficiency of the net. This is the ratio of the amount of information that can be retrieved from the net to the number of bits of storage available and is given by $\eta_o = R_o I_o / Z N_A N_B$, where $R_o$ is the capacity, $I_o$ is the amount of information contained in an output pattern and $Z N_A N_B$ is the number of weights, or bits of storage required (Willshaw et al., 1969; Buckingham & Willshaw, 1992). Information efficiency as a function of connectivity is shown in Figure 3. There is a distinct peak in information efficiency for each of the recall strategies at some low level of connectivity. The peak information efficiencies and the efficiencies at full connectivity are summarised in Table 2. The greatest contrast between full and partial connectivity is seen with the *normalised WTA* and noise-free cues. At 1% connectivity the *normalised WTA* is nearly 14 times more efficient than at full connectivity. In absolute terms, however, the *normalised* capacity is only 694 at 1% connectivity, compared with 5122 at full connectivity. The peak efficiency of 53% obtained by the *normalised WTA* is approaching the theoretically approximate maximum of 69% for a fully connected net (Willshaw et al., 1969).

## 5   DISCUSSION

Previous simulations (Graham & Willshaw, 1994) have shown that, when the input cues are noisy, the recall performance of the *winners-take-all* thresholding strategy applied to the partially connected associative net is greatly improved if the dendritic sums of the output units are transformed by functions of input activity and unit usage. We have confirmed and extended these results here by calculating the theoretical capacity of the associative net as a function of connectivity.

For the moderately sparse patterns used, all of the recall strategies are most information efficient at very low levels of connectivity ($\leq$ 10%). However, the optimum connectivity level is dependent on the pattern coding rate. Extending the analysis of Willshaw et al. (1969) to a partially connected net using *normalised WTA* recall yields that maximum information efficiency is obtained when $Z M_A = \log_2(N_B)$. So for input coding rates higher than $\log_2(N_B)$, a partially connected net is most information efficient. For the input coding rate used here, this relationship gives an optimum connectivity level of 0.87%, very close to the 1% obtained from the recall theory.

Comparing the peak efficiencies across the different strategies for the noisy cue case, the *normalised WTA* is about twice as efficient as the *basic WTA*, while the *transformed WTA* is three times as efficient. This comparison does not include the

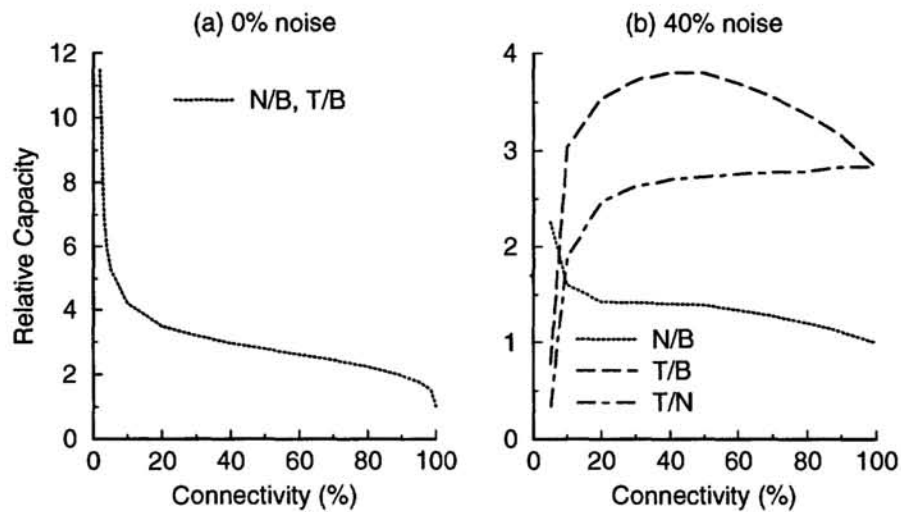

Figure 2: Relative Capacity Versus Connectivity

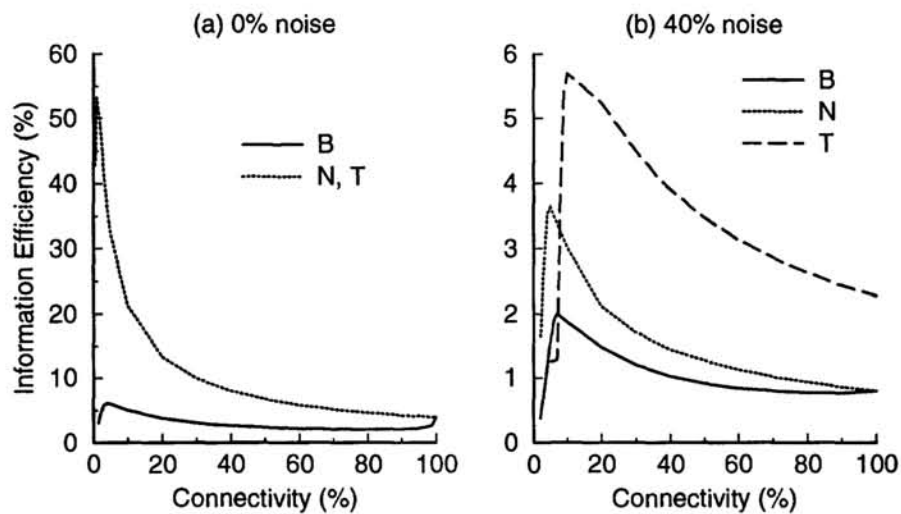

Figure 3: Information Efficiency Versus Connectivity

Table 2: Information Efficiency

| WTA Strategy | 0% Noise | | | 40% Noise | | |
|---|---|---|---|---|---|---|
| | $\eta_o$ at Peak (%) | $Z$ at Peak (%) | $\eta_o$ at $Z = 1$ (%) | $\eta_o$ at Peak (%) | $Z$ at Peak (%) | $\eta_o$ at $Z = 1$ (%) |
| Basic | 6.1 | 4 | 3.9 | 2.0 | 7 | 0.8 |
| Normalised | 53.3 | 1 | 3.9 | 3.6 | 5 | 0.8 |
| Transformed | 53.3 | 1 | 3.9 | 5.7 | 10 | 2.3 |

cost of storing input activity and unit usage information. If one bit of storage per connection is required for the input activity, and another bit for the unit usage, then the information efficiency of the *normalised WTA* is halved, and the information efficiency of the *transformed WTA* is reduced by two thirds. This results in all the strategies having about the same peak efficiency. However, the absolute capacities of the different strategies at their peak efficiencies are 183, 237 and 741 for the *basic*, *normalised* and *transformed WTA*, respectively. So, at the same level of efficiency, the *transformed WTA* delivers four times the capacity of the *basic WTA*.

In conclusion, numerical calculations of the capacity of the associative net show that it is most information efficient at a very low level of connectivity when moderately sparse patterns are stored. Including input activity and unit usage information into the recall calculations results in a four-fold increase in storage capacity without loss of efficiency.

### Acknowledgements

To the Medical Research Council for financial support under Programme Grant PG 9119632

# References

Buckingham, J., & Willshaw, D. (1992). Performance characteristics of the associative net. *Network, 3*, 407–414.

Buckingham, J., & Willshaw, D. (1993). On setting unit thresholds in an incompletely connected associative net. *Network, 4*, 441–459.

Buckingham, J. (1991). *Delicate nets, faint recollections: a study of partially connected associative network memories*. Ph.D. thesis, University of Edinburgh.

Graham, B., & Willshaw, D. (1994). Improving recall from an associative memory. *Biol. Cybern.*, in press.

Marr, D. (1971). Simple memory: a theory for archicortex. *Phil. Trans. Roy. Soc. Lond. B, 262*, 23–81.

Shepherd, G. (Ed.). (1990). *The Synaptic Organization of the Brain* (Third edition). Oxford University Press, New York, Oxford.

Willshaw, D. (1971). *Models of distributed associative memory*. Ph.D. thesis, University of Edinburgh.

Willshaw, D., Buneman, O., & Longuet-Higgins, H. (1969). Non-holographic associative memory. *Nature, 222*, 960–962.